# Experiences with Bayesian Learning in a Real World Application

**Peter Sykacek, Georg Dorffner**
Austrian Research Institute for Artificial Intelligence
Schottengasse 3, A-1010 Vienna Austria
peter, georg@ai.univie.ac.at

**Peter Rappelsberger**
Institute for Neurophysiology at the University Vienna
Währinger Straße 17, A-1090 Wien
Peter.Rappelsberger@univie.ac.at

**Josef Zeitlhofer**
Department of Neurology at the AKH Vienna
Währinger Gürtel 18-20, A–1090 Wien
Josef.Zeitlhofer@univie.ac.at

## Abstract

This paper reports about an application of Bayes' inferred neural network classifiers in the field of automatic sleep staging. The reason for using Bayesian learning for this task is two-fold. First, Bayesian inference is known to embody regularization automatically. Second, a side effect of Bayesian learning leads to larger variance of network outputs in regions without training data. This results in well known moderation effects, which can be used to detect outliers. In a 5 fold cross-validation experiment the full Bayesian solution found with R. Neals hybrid Monte Carlo algorithm, was not better than a single maximum a-posteriori (MAP) solution found with D.J. MacKay's evidence approximation. In a second experiment we studied the properties of both solutions in rejecting classification of movement artefacts.

# 1 Introduction

Sleep staging is usually based on rules defined by Rechtschaffen and Kales (see [8]). Rechtschaffen and Kales rules define 4 sleep stages, stage one to four, as well as rapid eye movement (REM) and wakefulness. In [1] J. Bentrup and S. Ray report that every year nearly one million US citizens consulted their physicians concerning their sleep. Since sleep staging is a tedious task (one all night recording on average takes about 3 hours to score manually), much effort was spent in designing automatic sleep stagers.

Sleep staging is a classification problem which was solved using classical statistical techniques or techniques emerged from the field of artificial intelligence (AI). Among classical techniques especially the k nearest neighbor technique was used. In [1] J. Bentrup and S. Ray report that the classical technique outperformed their AI approaches. Among techniques from the field of AI, researchers used inductive learning to build tree based classifiers (e.g. ID3, C4.5) as reported by M. Kubat et. al. in [4]. Neural networks have also been used to build a classifier from training examples. Among those who used multi layer perceptron networks to build the classifier, the work of R. Schaltenbrand et. al. seems most interesting. In [10] they use a separate network to refuse classification of too distant input vectors. The performance usually reported is in the range of 75 to 85 percent.

Which enhancements to these approaches can be made to get a reliable system with hopefully better performance? According to S. Roberts et. al. in [9], outlier detection is important to get reliable results in a critical (e.g. medical) environment. To get reliable results one must refuse classification of dubious inputs. Those inputs are marked separately for further inspection by a human expert. To be able to detect such dubious inputs, we use Bayesian inference to calculate a distribution over the neural network weights. This approach automatically incorporates the calculation of confidence for each network estimate. Bayesian inference has the further advantage that regularization is part of the learning algorithm. Additional methods like weight decay penalty and cross validation for decay parameter tuning are no longer needed. Bayesian inference for neural networks was among others investigated by D.J. MacKay (see [5]), Thodberg (see [11]) and Buntine and Weigend (see [3]).

The aim of this paper is to study how Bayesian inference leads to probabilities for classes, which together with doubt levels allow to refuse classification of outliers. As we are interested in evaluating the resulting performance, we use a comparative method on the same data set and use a significance test, such that the effect of the method can easily be evaluated.

# 2 Methods

In this section we give a short description of the inference techniques used to perform the experiments. We have used two approaches using neural networks as classifiers and an instance based approach in order to make the performance estimates comparable to other methods.

## 2.1 Architecture for polychotomous classification

For polychotomous classification problems usually a 1-of-c target coding scheme is used. Usually it is sufficient to use a network architecture with one hidden layer. In [2] pp. 237–240, C. Bishop gives a general motivation for the softmax data model,

which should be used if one wants the network outputs to be probabilities for classes.

If we assume that the class conditional densities, $p(\underline{z} \mid C_k)$, of the hidden unit activation vector, $\underline{z}$, are from the general family of exponential distributions, then using the transformation in (1), allows to interpret the network outputs as probabilities for classes. This transformation is known as normalized exponential or softmax activation function.

$$p(C_k \mid \underline{z}) = \frac{\exp(a_k)}{\sum_{k'} \exp(a_{k'})} \tag{1}$$

In (1) the value $a_k$ is the value at output node $k$ before applying softmax activation. Softmax transformation of the activations in the output layer is used for both network approaches used in this paper.

## 2.2 Bayesian Inference

In [6] D.J. MacKay uses Bayesian inference and marginalization to get moderated probabilities for classes in regions where the network is uncertain about the class label. In conjunction with doubt levels this allows to suppress a classification of such patterns. A closer investigation of this approach showed that marginalization leads to moderated probabilities, but the degree of moderation heavily depends on the direction in which we move away from the region with sufficient training data. Therefore one has to be careful about whether the moderation effect should be used for outliers detection.

A Bayesian solution for neural networks is a posterior distribution over weight space calculated via Bayes' theorem using a prior over weights.

$$p(\underline{w} \mid \mathcal{D}) = \frac{p(\mathcal{D} \mid \underline{w})p(\underline{w})}{p(\mathcal{D})} \tag{2}$$

In (2), $\underline{w}$ is the weight vector of the network and $\mathcal{D}$ represents the training data. Two different possibilities are known to calculate the posterior in (2). In [5] D.J. MacKay derives an analytical expression assuming a Gaussian distribution. In [7] R. Neal uses a hybrid Monte Carlo method to sample from the posterior. For one input pattern, the posterior over weight space will lead to a distribution of network outputs.

For a classification problem, following MacKay [6], the network estimate is calculated by marginalization over the output distribution.

$$P(C_1 \mid \underline{x}, \mathcal{D}) = \int P(C_1 \mid \underline{x}, \underline{w})p(\underline{w} \mid \mathcal{D})d\underline{w}$$
$$= \int y(\underline{x}, \underline{w})p(\underline{w} \mid \mathcal{D})d\underline{w} \tag{3}$$

In general, the distribution over output activations will have small variance in regions well represented in the training data and large variance everywhere else. The reason for that is the influence of the likelihood term $p(\mathcal{D} \mid \underline{w})$, which forces the network mapping to lie close to the desired one in regions with training data, but which has no influence on the network mapping in regions without training data. At least for for generalized linear models applied to regression, this property is quantifiable. In [12] C. Williams et.al. showed that the error bar is proportional to the inverse input data density $p(\underline{x})^{-1}$. A similar relation is also plausible for the output activation in classification problems.

Due to the nonlinearity of the softmax transformation, marginalization will moderate probabilities for classes. Moderation will be larger in regions with large variance of the output activation. Compared to a decision made with the most probable weight, the network guess for the class label will be less certain. This moderation effect allows to reject classification of outlying patterns.

Since upper integral can not be solved analytically for classification problems, there are two possibilities to solve it. In [6] D.J. MacKay uses an approximation. Using hybrid Monte Carlo sampling as an implementation of Bayesian inference (see R. Neal in [7]), there is no need to perform upper integration analytically. The hybrid Monte Carlo algorithm samples from the posterior and upper integral is calculated as a finite sum.

$$P(C_1 \mid \underline{x}, \mathcal{D}) \approx \frac{1}{L} \sum_{i=1}^{L} y(\underline{x}, \underline{w}_i) \tag{4}$$

Assuming, that the posterior over weights is represented exactly by the sampled weights, there is no need to limit the number of hidden units, if a correct (scaled) prior is used. Consequently in the experiments the network size was chosen to be large. We used 25 hidden units. Implementation details of the hybrid Monte Carlo algorithm may be found in [7].

## 2.3 The Competitor

The classifier, used to give performance estimates to compare to, is built as a two layer perceptron network with softmax transformation applied to the outputs. As an error function we use the cross entropy error including a consistent weight decay penalty, as it is e.g. proposed by C. Bishop in [2], pp. 338. The decay parameters are estimated with D.J. MacKay's evidence approximation ( see [5] for details). Note that the restriction of D.J. MacKay's implementation of Bayesian learning, which has no solution to arrive at moderated probabilities in 1-of-c classification problems, do not apply here since we use only one MAP value. The key problem with this approach is the Gaussian approximation of the posterior over weights, which is used to derive the most probable decay parameters. This approximation is certainly only valid if the number of network parameters is small compared to the number of training samples. One consequence is, that the size of the network has to be restricted. Our model uses 6 hidden units.

To make the performance of the Bayes inferred classifier also comparable to other methods, we decided to include performance estimates of a k nearest neighbor algorithm. This algorithm is easy to implement and from [1] we have some evidence that its performance is good.

## 3 Experiments and Results

In this section we discuss the results of a sleep staging experiment based on the techniques described in the "Methods" section.

### 3.1 Data

All experiments are performed with spectral features calculated from a database of 5 different healthy subjects. All recordings were scored according to the Rechtschaffen & Kales rules. The data pool consisted from data calculated for all electrodes

available, which were horizontal eye movement, vertical eye movement and 18 EEG electrodes placed with respect to the international 10-20 system.

The data were transformed into the frequency domain. We used power density values as well as coherency between different electrodes, which is a correlation coefficient expressed as a function of frequency as input features. All data were transformed to zero mean and unit variance. From the resulting feature space we selected 10 features, which were used as inputs for classification. Feature selection was done with a suboptimal search algorithm which used the performance of a k nearest neighbor classifier for evaluation. We used more than 2300 samples during training and about 580 for testing.

## 3.2    Analysis of Both Classifiers

The analysis of both classifiers described in the "Methods" section should reveal whether besides good classification performance the Bayes' inferred classifier is also capable of refusing outlying test patterns. Increasing the doubt level should lead to better results of the classifier trained by Bayesian Inference if the test data contains outlying patterns. We performed two experiments. During the first experiment we calculated results from a 5 fold cross validation, where training is done with 4 subjects and tests are performed with one independent test person. In a second test we examine the differences of both algorithms on patterns which are definitely outliers. We used the same classifiers as in the first experiment. Test patterns for this experiment were classified movement artefacts, which should not be classified as one of the sleep stages.

The classifier used in conjunction with Bayesian inference was a 2-layer neural network with 10 inputs, 25 hidden units with sigmoid activation and five output units with softmax activation. The large number of hidden units is motivated by the results reported from R. Neal in [7]. R. Neal studied the properties of neural networks in a Bayesian framework when using Gaussian priors over weights. He concluded that there is no need for limiting the complexity of the network when using a correct Bayesian approach. The standard deviation of the Gaussian prior is scaled by the number of hidden units.

For the comparative approach we used a neural network with 10 inputs, 6 hidden units and 5 outputs with softmax activation. Optimization was done via the BFGS algorithm (see C. Bishop in [2]) with automatic weight decay parameter tuning (D.J. MacKay's evidence approximation). As described in the methods section, the smaller network used here is motivated by the Gaussian approximation of the posterior over weights, which is used in the expression for the most probable decay parameters.

The third result is a result achieved with a k nearest neighbor classifier with k set to three.

All results are summaried in table 1. Each column summarizes the results achieved with one of the algorithms and a certain doubt level during the cross validation run. As the k nearest neighbor classifier gives only coarse probability estimates, we give only the performance estimate when all test patterns are classified.

An examination of table 1 shows that the differences between the MAP-solution and the Bayesian solution are extremely small. Consequently, using a t-test, the 0-hypothesis could not be rejected at any reasonable significance level. On the other hand compared to the Bayesian solution, the performance of the k nearest neighbor classifier is significantly lower (the significance level is 0.001).

Table 1: Classification Performance

| MAP | | | | |
|---|---|---|---|---|
| Doubt Cases | 0 | 5% | 10% | 15% |
| Mean Perf. | 78.6% | 80.4% | 81.6% | 83.2% |
| Std. Dev. | 9.1% | 9.4% | 9.4% | 9.7% |
| Bayes | | | | |
| Doubt Cases | 0 | 5% | 10% | 15% |
| Mean Perf. | 78.4% | 80.2% | 82.2% | 83.6% |
| Std. Dev. | 8.6% | 9.0% | 9.4% | 9.7% |
| k nearest neighbor | | | | |
| Doubt Cases | 0 | 5% | 10% | 15% |
| Mean Perf. | 74.6% | - | - | - |
| Std. Dev. | 8.4% | - | - | - |

Table 2: Rejection of Movement Periods

| Method | MAP | | Bayes | |
|---|---|---|---|---|
| recognized outliers | No. | % | No. | % |
| | 0 | 0% | 1 | 7.7% |
| | 1 | 7.7% | 6 | 46.1% |
| | 2 | 15.4% | 5 | 38.5% |
| | 0 | 0% | 5 | 38.5% |
| | 1 | 7.7% | 3 | 23.1% |

The last experiment revealed that both training algorithms lead to comparable performance estimates, when clean data is used. When using the classifier in practice there is no guarantee that the data are clean. One common problem of all night recordings are the so called movement periods, which are periods with muscle activity due to movements of the sleeping subject. During a second experiment we tried to assess the robustness of both neural classifiers against such inputs. During this experiment we used a fixed doubt level, for which approximately 5% of the clean test data from the last experiment were rejected. With this doubt level we classified 13 movement periods, which should not be assigned to any of the other stages. The number of correctly refused outlying patterns are shown in table 2. Analysis of the results with a t-test showed a significant higher rate of removed outliers for the full Bayesian approach. Nevertheless as the number of misclassified outliers is large, one has to be careful in using this side-effect of Bayesian inference.

## 4 Conclusion

Using Bayesian Inference for neural network training is an approach which leads to better classification results compared with simpler training procedures. Comparing with the "one MAP" solution, we observed significantly larger reliability in detecting dubious patterns. The large amount of remaining misclassified patterns, which were obviously outlying, shows that we should not rely blindly on the moderating effect of marginalization. Despite the large amount of time which is required to calculate the solution, Bayesian inference has relevance for practical applications. On one hand the Bayesian solution shows good performance. But the main reason is the ability to encode a validity region of the model into the solution. Compared to all methods which do not aim at a predictive distribution, this is a clear advantage for Bayesian inference.

## Acknowledgements

We want to acknowledge the work of R. Neal from the Departments of Statistics and Computer Science at the University of Toronto, who made his implementation of hybrid Monte-Carlo sampling for Bayesian inference available electronically. His software was used to calculate the full Bayes' inferred classification results. We also want to express gratitude to S. Roberts from Imperial College London, one of the partners in the ANNDEE project. His work and his consequence in insisting on confidence measures for network decisions had a large positive impact on our work.

This work was sponsored by the Austrian Federal Ministry of Science and Transport. It was done in the framework of the BIOMED 1 concerted action ANNDEE, financed by the European Commission, DG. XII.

## References

[1] J.A. Bentrup and S.R. Ray. An examination of inductive learning algorithms for the classification of sleep signals. Technical Report UIUCDCS-R-93-1792, Dept of Computer Science, University of Illinois, Urbana-Champaign, 1993.

[2] C. M. Bishop. *Neural Networks for Pattern Recognition*. Clarendon Press, Oxford, 1995.

[3] W. L. Buntine and A. S. Weigend. Bayesian back-propagation. *Complex Systems*, 5:603–643, 1991.

[4] M. Kubat, G. Pfurtscheller, and D. Flotzinger. Discrimination and classification using both binary and continuous variables. *Biological Cybernetics*, 70:443–448, 1994.

[5] D. J. C. MacKay. Bayesian interpolation. *Neural Computation*, 4:415–447, 1992.

[6] D. J. C. MacKay. The evidence framework applied to classification networks. *Neural Computation*, 4:720–736, 1992.

[7] R. M. Neal. *Bayesian Learning for Neural Networks*. Springer, New York, 1996.

[8] A. Rechtschaffen and A. Kales. *A manual of standardized terminology, techniques and scoring system for sleep stages of human subjects*. NIH Publication No. 204, US Government Printing Office, Washington, DC., 1968.

[9] S. Roberts, L. Tarassenko, J. Pardey, and D. Siegwart. A confidence measure for artificial neural networks. In *International Conference Neural Networks and Expert Systems in Medicine and Healthcare*, pages 23–30, Plymouth, UK, 1994.

[10] N. Schaltenbrand, R. Lengelle, and J.P. Macher. Neural network model: application to automatic analysis of human sleep. *Computers and Biomedical Research*, 26:157–171, 1993.

[11] H. H. Thodberg. A review of bayesian neural networks with an application to near infrared spectroscopy. *IEEE Transactions on Neural Networks*, 7(1):56–72, January 1996.

[12] C. K. I. Williams, C. Quazaz, C. M. Bishop, and H. Zhu. On the relationship between bayesian error bars and the input data density. In *Fourth International Conference on Artificial Neural Networks, Churchill College, University of Cambridge, UK. IEE Conference Publication No. 409*, pages 160–165, 1995.